# Dynamically Adapting Kernels in Support Vector Machines

**Nello Cristianini**
Dept. of Engineering Mathematics
University of Bristol, UK
nello.cristianini@bristol.ac.uk

**Colin Campbell**
Dept. of Engineering Mathematics
University of Bristol, UK
c.campbell@bristol.ac.uk

**John Shawe-Taylor**
Dept. of Computer Science
Royal Holloway College
john@dcs.rhbnc.ac.uk

## Abstract

The kernel-parameter is one of the few tunable parameters in Support Vector machines, controlling the complexity of the resulting hypothesis. Its choice amounts to model selection and its value is usually found by means of a validation set. We present an algorithm which can automatically perform model selection with little additional computational cost and with no need of a validation set. In this procedure model selection and learning are not separate, but kernels are dynamically adjusted during the learning process to find the kernel parameter which provides the best possible upper bound on the generalisation error. Theoretical results motivating the approach and experimental results confirming its validity are presented.

## 1  Introduction

Support Vector Machines (SVMs) are learning systems designed to automatically trade-off accuracy and complexity by minimizing an upper bound on the generalisation error provided by VC theory. In practice, however, SVMs still have a few tunable parameters which need to be determined in order to achieve the right balance and the values of these are usually found by means of a validation set. One of the most important of these is the kernel-parameter which implicitly defines the structure of the high dimensional feature space where the maximal margin hyperplane is found. Too rich a feature space would cause the system to overfit the data,

and conversely the system can be unable to separate the data if the kernels are too poor. Capacity control can therefore be performed by tuning the kernel parameter subject to the margin being maximized. For noisy datasets, yet another quantity needs to be set, namely the soft-margin parameter $C$.

SVMs therefore display a remarkable dimensionality reduction for model selection. Systems such as neural networks need many different architectures to be tested and decision trees are faced with a similar problem during the pruning phase. On the other hand SVMs can shift from one model complexity to another by simply tuning a continuous parameter.

Generally, model selection by SVMs is still performed in the standard way: by learning different SVMs and testing them on a validation set in order to determine the optimal value of the kernel-parameter. This is expensive in terms of computing time and training data. In this paper we propose a different scheme which dynamically adjusts the kernel-parameter to explore the space of possible models at little additional computational cost compared to fixed-kernel learning. Futhermore this approach only makes use of training-set information so it is more efficient in a sample complexity sense.

Before proposing the model selection procedure we first prove a theoretical result, namely that the margin and structural risk minimization (SRM) bound on the generalization error depend smoothly on the kernel parameter. This can be exploited by an algorithm which keeps the system close to maximal margin while the kernel parameter is changed smoothly. During this phase, the theoretical bound given by SRM theory can be computed. The best kernel-parameter is the one which gives the lowest possible bound. In section 4 we present experimental results showing that model selection can be efficiently performed using the proposed method (though we only consider Gaussian kernels in the simulations outlined).

## 2   Support Vector Learning

The decision function implemented by SV machines can be written as:

$$f(x) = \text{sign}\left(\sum_{i \in \text{SV}} y_i \alpha_i^o K(x, x_i) - \theta\right)$$

where the $\alpha_i^o$ are obtained by maximising the following Lagrangian (where $m$ is the number of patterns):

$$L = \sum_{i=1}^{m} \alpha_i - 1/2 \sum_{i,j=1}^{m} \alpha_i \alpha_j y_i y_j K(x_i, x_j)$$

with respect to the $\alpha_i$, subject to the constraints

$$\alpha_i \geq 0 \qquad \sum_{i=1}^{m} \alpha_i y_i = 0$$

and where the functions $K(x, x')$ are called *kernels*. The kernels provide an expression for dot-products in a high-dimensional *feature space* [1]:

$$K(x, x') = \langle \Phi(x), \Phi(x') \rangle$$

and also implicitly define the nonlinear mapping $\Phi(x)$ of the training data into feature space where they may be separated using the maximal margin hyperplane. A number of choices of kernel-function can be made e.g. Gaussians kernels:

$$K(x, x') = e^{-\|x-x'\|^2/2\sigma^2}$$

The following upper bound can be proven from VC theory for the generalisation error using hyperplanes in feature space [7, 9]:

$$\epsilon \leq O\left(\frac{R^2}{m\gamma^2}\right)$$

where $R$ is the radius of the smallest ball containing the training set, $m$ the number of training points and $\gamma$ the margin (cf. [2] for a complete survey of the generalization properties of SV machines).

The Lagrange multipliers $\alpha_i$ are usually found by means of a Quadratic Programming optimization routine, while the kernel-parameters are found using a validation set. As illustrated in Figure 1 there is a minimum of the generalisation error for that value of the kernel-parameter which has the best trade-off between overfitting and ability to find an efficient solution.

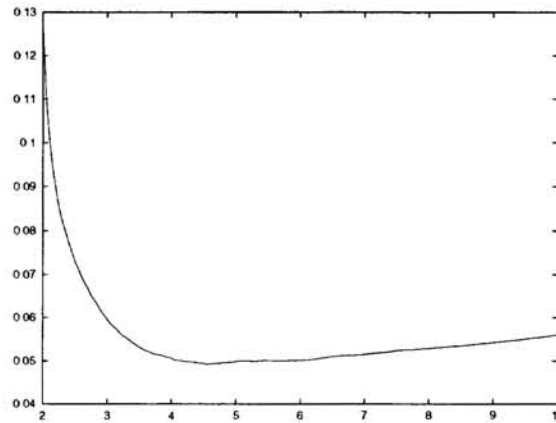

Figure 1: Generalization error ($y$-axis) as a function of $\sigma$ ($x$-axis) for the mirror symmetry problem (for Gaussian kernels with zero training error and maximal margin, $m = 200$, $n = 30$ and averaged over $10^5$ examples).

## 3   Automatic Model Order Selection

We now prove a theorem which shows that the margin of the optimal hyperplane is a smooth function of the kernel parameter, as is the upper bound on the generalisation error. First we state the Implicit Function Theorem.

**Implicit Function Theorem [10]:** Let $\overline{F}(x, \overline{y})$ be a continuously differentiable function,

$$\overline{F} : U \subseteq \Re \times V \subseteq \Re^p \to \Re$$

and let $(a, \overline{b}) \in U \times V$ be a solution to the equation $\overline{F}(x, \overline{y}) = 0$. Let the partial derivatives matrix $m_{i,j} = (\frac{\partial F_i}{\partial y_j})$ w.r.t. $y$ be full rank at $(a, \overline{b})$. Then, near $(a, \overline{b})$,

there exists one and only one function $\overline{y} = \overline{g}(x)$ such that $\overline{F}(x, \overline{g}(x)) = 0$, and such function is continuous.

**Theorem:** The *margin* $\gamma$ of SV machines depends smoothly on the kernel parameter $\sigma$.

**Proof:** Consider the function $\overline{g} : \Sigma \subseteq \Re \to A \subseteq \Re^p, \overline{g} : \sigma \mapsto (\overline{\alpha}^o, \lambda)$ which given the data maps the choice of $\sigma$ to the optimal parameters $\alpha^o$ and lagrange parameter $\lambda$ of the SV machine with Kernel matrix $G_{ij} = y_i y_j K(\sigma; x_i, x_j))$. Let

$$W_\sigma(\overline{\alpha}) = \sum_{i=1}^p \overline{\alpha}_i - 1/2 \sum_{i,j} \overline{\alpha}_i \overline{\alpha}_j y_i y_j K(\sigma; x_i, x_j) + \lambda(\sum_i y_i \overline{\alpha}_i)$$

be the functional that the SV machine maximizes. Fix a value of $\sigma$ and let $\alpha^o(\sigma)$ be the corresponding solution of $W_\sigma(\overline{\alpha})$. Let $I$ be the set of indices for which $\overline{\alpha}_j^o(\sigma) \neq 0$. We may assume that the submatrix of $G$ indexed by $I$ is non-singular since otherwise the maximal margin hyperplane could be expressed in terms of a subset of indices. Now choose a maximal set of indices $J$ containing $I$ such that the corresponding submatrix of $G$ is non-singular and all of the points indexed by $J$ have margin 1. Now consider the function $F(\sigma, \overline{\alpha}, \lambda)_i = \left(\frac{\partial W_\sigma}{\partial \alpha}\right)_{j_i}, i \geq 1, F(\sigma, \overline{\alpha}, \lambda)_0 = \sum_j y_j \overline{\alpha}_j$ in the neighbourhood of $\sigma$, where $j_i$ is an enumeration of the elements of $J$,

$$\frac{\partial W_\sigma}{\partial \alpha_j} = 1 - y_j \sum_i \overline{\alpha}_i y_i K(\sigma; x_i, x_j) + \lambda y_j$$

and satisfies the equation $F(\sigma, \overline{\alpha^o}(\sigma), \lambda(\sigma)) = \mathbf{0}$ at the extremal points of $W_\sigma(\overline{\alpha})$. Then the SV function is the implicit function, $(\overline{\alpha}^o, \lambda) = \overline{g}(\sigma)$, and is continuous (and unique) *iff* $\overline{F}$ is continuously differentiable and the partial derivatives matrix w.r.t. $\overline{\alpha}, \lambda$ is full rank. But the partial derivatives matrix $H$ is given by

$$H_{ij} = \frac{\partial F_i}{\partial \alpha_{j_j}} = y_{j_i} y_{j_j} K(\sigma; x_{j_i}, x_{j_j}) = H_{ji}, i, j \geq 1,$$

for $j_i, j_j \in J$, which was non-degenerate by definition of $J$, while

$$H_{00} = \frac{\partial F_0}{\partial \lambda} = 0 \quad \text{and} \quad H_{0j} = \frac{\partial F_0}{\partial \alpha_{j_j}} = y_{j_j} = \frac{\partial F_j}{\partial \lambda} = H_{j0}, j \geq 1.$$

Consider any non-zero $\alpha$ satisfying $\sum_j \alpha_j y_j = 0$, and any $\lambda$. We have

$$(\alpha, \lambda)^T H(\alpha, \lambda) = \alpha^T G\alpha + 2\lambda \alpha^T y = \alpha^T G\alpha > 0.$$

Hence, the matrix $H$ is non-singular for $\alpha$ satisfying the given linear constraint. Hence, by the implicit function theorem $g$ is a continuous function of $\sigma$. The following is proven in [2]:

$$\gamma^2 = \left(\sum_{i=1}^p \overline{\alpha}_i^o\right)^{-1}$$

which shows that $\gamma$ is a continuous function of $\sigma$. As the radius of the ball containing the points is also a continuous function of $\sigma$, and the generalization error bound has the form $\epsilon \leq CR(\sigma)^2 \|\alpha^o(\sigma)\|_1$ for some constant $C$, we have the following corollary.

**Corollary:** The bound on the generalization error is smooth in $\sigma$.

This means that, when the margin is optimal, small variations in the kernel parameter will produce small variations in the margin (and in the bound on the generalisation error). Thus $\gamma_\sigma \approx \gamma_{\sigma+\delta\sigma}$ and after updating the $\sigma$, the system will

still be in a sub-optimal position. This suggests the following strategy for Gaussian kernels, for instance:

**Kernel Selection Procedure**

1. Initialize $\sigma$ to a very small value

2. Maximize the margin, then

   - Compute the SRM bound (or observe the validation error)
   - Increase the kernel parameter: $\sigma \leftarrow \sigma + \delta\sigma$

3. Stop when a predetermined value of $\sigma$ is reached else repeat step 2.

This procedure takes advantage of the fact that for very small $\sigma$ convergence is generally very rapid (overfitting the data, of course), and that once the system is near the equilibrium, few iterations will always be sufficient to move it back to the maximal margin situation. In other words, this system is brought to a maximal margin state in the beginning, when this is computationally very cheap, and then it is actively kept in that situation by continuously adjusting the $\alpha$ while the kernel-parameter is gradually increased.

In the next section we will experimentally investigate this procedure for real-life datasets. In the numerical simulations we have used the Kernel-Adatron (KA) algorithm recently developed by two of the authors [4] which can be used to train SV machines. We have chosen this algorithm because it can be regarded as a gradient ascent procedure for maximising the Kuhn-Tucker Lagrangian $L$. Thus the $\alpha_i$ for a sub-optimal state are close to those for the optimum and so little computational effort will be needed to bring the system back to a maximal margin position:

**The Kernel-Adatron Algorithm.**

1. $\alpha_i = 1$.

2. FOR $i = 1$ TO m

   - $z_i = \sum_{j=1}^{m} \alpha_j y_j K(x_i, x_j)$
   - $\gamma_i = y_i z_i$
   - $\delta\alpha^i = \eta(1 - \gamma^i)$
   - IF $(\alpha^i + \delta\alpha^i) \leq 0$ THEN $\alpha^i = 0$ ELSE $\alpha^i \leftarrow \alpha^i + \delta\alpha^i$.
   - $margin = \frac{1}{2}\left(\min\left(z_i^+\right) - \max\left(z_i^-\right)\right)$
     ($z_i^+$ ($z_i^-$) = positively (negatively) labelled patterns)

3. IF($margin = 1$) THEN stop, ELSE go to step 2.

## 4  Experimental Results

In this section we implement the above algorithm for real-life datasets and plot the upper bound given by VC theory and the generalization error as functions of $\sigma$. In order to compute the bound, $\epsilon \leq R^2/m\gamma^2$ we need to estimate the radius of the ball in feature space. In general his can be done explicitly by maximising the following Lagrangian w.r.t. $\lambda_i$ using convex quadratic programming routines:

$$L = \sum_i \lambda_i K(x_i, x_i) - \sum_{i,j} \lambda_i \lambda_j K(x_i, x_j)$$

subject to the constraints $\sum_i \lambda_i = 1$ and $\lambda_i \geq 0$. The radius is then found from [3]:

$$R = \sum_{i,j} \lambda_i \lambda_j K(x_i, x_j) - 2 \sum_{i,j} \lambda_j K(x_i, x_j) + \sum_i K(x_i, x_i)$$

However, we can also get an upper bound for this quantity by noting that Gaussian kernels always map training points to the surface of a sphere of radius 1 centered on the origin of the feature space. This can be easily seen by noting that the distance of a point from the origin is its norm:

$$||\Phi(x)|| = \sqrt{\langle \Phi(x), \Phi(x) \rangle} = \sqrt{K(x,x)} = \sqrt{e^{||x-x||/2\sigma^2}} = 1$$

In Figure 2 we give both these bounds (the upper bound is $\sum_i \alpha_i / m$) and generalisation error (on a test set) for two standard datasets: the aspect-angle dependent sonar classification dataset of Gorman and Sejnowski [5] and the Wisconsin breast cancer dataset [8]. As we see from these plots there is little need for the additional computational cost of determining $R$ from the above quadratic progamming problem, at least for Gaussian kernels. In Fig. 3 we plot the bound $\sum_i \alpha_i / m$ and generalisation error for 2 figures from a United States Postal Service dataset of handwritten digits [6]. In these, and other instances we have investigated, the minimum of the bound approximately coincides with the minimum of the generalisation error. This gives a good criterion for the most suitable choice for $\sigma$. Furthermore, this estimate for the best $\sigma$ is derived solely from training data without the need for an additional validation set.

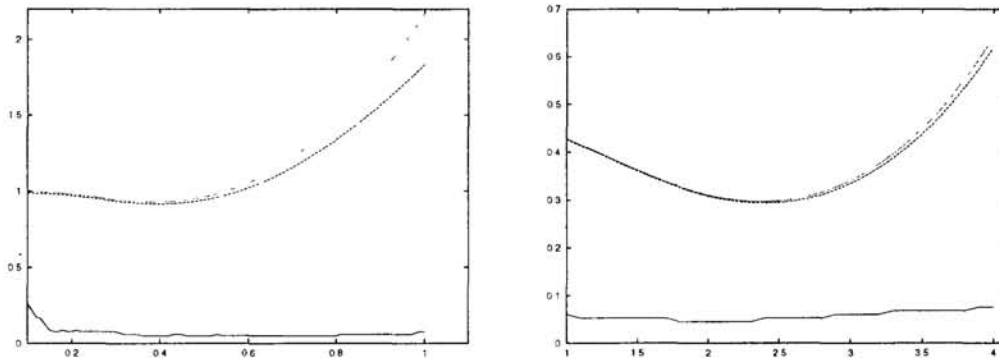

Figure 2: Generalisation error (solid curves) for the sonar classification (left Fig.) and Wisconsin breast cancer datasets (right Fig.). The upper curves (dotted) show the upper bounds from VC theory (for the top curves R=1).

Starting with a small $\sigma$-value we have observed that the margin can be maximised rapidly. Furthermore, the margin remains close to 1 if $\sigma$ is incremented by a small amount. Consequently, we can study the performance of the system by traversing a range of $\sigma$-values, alternately incrementing $\sigma$ then maximising the margin using the previous optimal set of $\alpha$-values as a starting point. We have found that this procedure does not add a significant computational cost in general. For example, for the sonar classification dataset mentioned above and starting at $\sigma = 0.1$ with increments $\Delta\sigma = 0.1$ it took 186 iterations to reach $\sigma = 1.0$ and 4895 to reach $\sigma = 2.0$ as against 110 and 2624 iterations for learning at both these $\sigma$-values. For a rough doubling of the learning time it is possible to determine a reasonable value for $\sigma$ for good generalisation without use of a validation set.

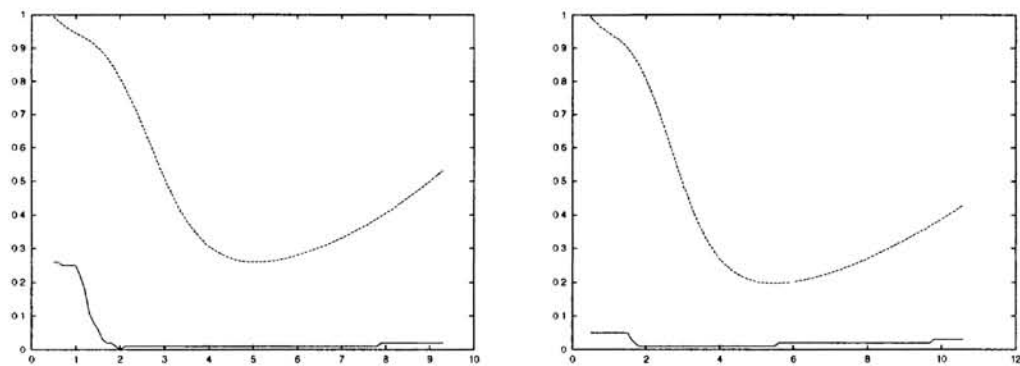

Figure 3: Generalisation error (solid curve) and upper bound from VC theory (dashed curve with R=1) for digits 0 and 3 from the USPS dataset of handwritten digits.

## 5 Conclusion

We have presented an algorithm which automatically learns the kernel parameter with little additional cost, both in a computational and sample-complexity sense. Model selection takes place during the learning process itself, and experimental results are provided showing that this strategy provides a good estimate of the correct model complexity.

## References

[1] Aizerman, M., Braverman, E., and Rozonoer, L. (1964). Theoretical Foundations of the Potential Function Method in Pattern Recognition Learning, *Automations and Remote Control*, **25**:821-837.

[2] Bartlett P., Shawe-Taylor J., (1998). Generalization Performance of Support Vector Machines and Other Pattern Classifiers. 'Advances in Kernel Methods - Support Vector Learning', Bernhard Schölkopf, Christopher J. C. Burges, and Alexander J. Smola (eds.), MIT Press, Cambridge, USA.

[3] Burges C., (1998). A tutorial on support vector machines for pattern recognition. *Data Mining and Knowledge Discovery*, **2**:1.

[4] Friess T., Cristianini N., Campbell C., (1998) The Kernel-Adatron Algorithm: a Fast and Simple Learning Procedure for Support Vector Machines, in Shavlik, J., ed., *Machine Learning: Proceedings of the Fifteenth International Conference*, Morgan Kaufmann Publishers, San Francisco, CA.

[5] Gorman R.P. & Sejnowski, T.J. (1988) *Neural Networks* **1**:75-89.

[6] LeCun, Y., Jackel, L. D., Bottou, L., Brunot, A., Cortes, C., Denker, J. S., Drucker, H., Guyon, I., Muller, U. A., Sackinger, E., Simard, P. and Vapnik, V., (1995). Comparison of learning algorithms for handwritten digit recognition, *International Conference on Artificial Neural Networks*, Fogelman, F. and Gallinari, P. (Ed.), pp. 53-60.

[7] Shawe-Taylor, J., Bartlett, P., Williamson, R. & Anthony, M. (1996). Structural Risk Minimization over Data-Dependent Hierarchies NeuroCOLT Technical Report NC-TR-96-053 (`ftp://ftp.dcs.rhbnc.ac.uk /pub/neurocolt/tech_reports`).

[8] Ster, B., & Dobnikar, A. (1996) Neural networks in medical diagnosis: comparison with other methods. In A. Bulsari et al. (ed.) *Proceedings of the International Conference EANN'96*, p. 427-430.

[9] Vapnik, V. (1995) *The Nature of Statistical Learning Theory*, Springer Verlag.

[10] James, Robert C. (1966) *Advanced calculus* Belmont, Calif. : Wadsworth
